# Rates of convergence for the cluster tree

**Kamalika Chaudhuri**
UC San Diego
kchaudhuri@ucsd.edu

**Sanjoy Dasgupta**
UC San Diego
dasgupta@cs.ucsd.edu

## Abstract

For a density $f$ on $\mathbb{R}^d$, a *high-density cluster* is any connected component of $\{x : f(x) \geq \lambda\}$, for some $\lambda > 0$. The set of all high-density clusters form a hierarchy called the *cluster tree* of $f$. We present a procedure for estimating the cluster tree given samples from $f$. We give finite-sample convergence rates for our algorithm, as well as lower bounds on the sample complexity of this estimation problem.

## 1 Introduction

A central preoccupation of learning theory is to understand what statistical estimation based on a finite data set reveals about the underlying distribution from which the data were sampled. For *classification* problems, there is now a well-developed theory of generalization. For *clustering*, however, this kind of analysis has proved more elusive.

Consider for instance $k$-means, possibly the most popular clustering procedure in use today. If this procedure is run on points $X_1, \ldots, X_n$ from distribution $f$, and is told to find $k$ clusters, what do these clusters reveal about $f$? Pollard [8] proved a basic consistency result: if the algorithm always finds the global minimum of the $k$-means cost function (which is NP-hard, see Theorem 3 of [3]), then as $n \to \infty$, the clustering is the globally optimal $k$-means solution for $f$. This result, however impressive, leaves the fundamental question unanswered: is the best $k$-means solution to $f$ an interesting or desirable quantity, in settings outside of vector quantization?

In this paper, we are interested in clustering procedures whose output on a finite sample converges to "natural clusters" of the underlying distribution $f$. There are doubtless many meaningful ways to define natural clusters. Here we follow some early work on clustering (for instance, [5]) by associating clusters with *high-density connected regions*. Specifically, a cluster of density $f$ is any connected component of $\{x : f(x) \geq \lambda\}$, for any $\lambda > 0$. The collection of all such clusters forms an (infinite) hierarchy called the *cluster tree* (Figure 1).

Are there hierarchical clustering algorithms which converge to the cluster tree? Previous theory work [5, 7] has provided weak consistency results for the single-linkage clustering algorithm, while other work [13] has suggested ways to overcome the deficiencies of this algorithm by making it more robust, but without proofs of convergence. In this paper, we propose a novel way to make single-linkage more robust, while retaining most of its elegance and simplicity (see Figure 3). We establish its finite-sample rate of convergence (Theorem 6); the centerpiece of our argument is a result on continuum percolation (Theorem 11). We also give a lower bound on the problem of cluster tree estimation (Theorem 12), which matches our upper bound in its dependence on most of the parameters of interest.

## 2 Definitions and previous work

Let $\mathcal{X}$ be a subset of $\mathbb{R}^d$. We exclusively consider Euclidean distance on $\mathcal{X}$, denoted $\| \cdot \|$. Let $B(x, r)$ be the closed ball of radius $r$ around $x$.

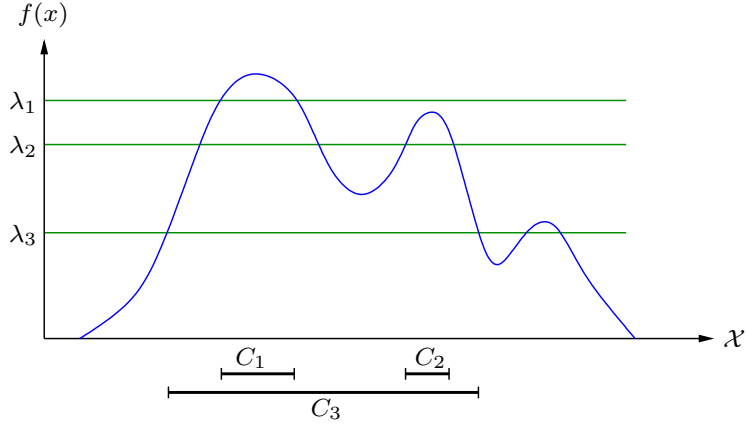

Figure 1: A probability density $f$ on $\mathbb{R}$, and three of its clusters: $C_1$, $C_2$, and $C_3$.

## 2.1 The cluster tree

We start with notions of connectivity. A *path* $P$ in $S \subset \mathcal{X}$ is a continuous $1 - 1$ function $P :$ $[0, 1] \to S$. If $x = P(0)$ and $y = P(1)$, we write $x \overset{P}{\rightsquigarrow} y$, and we say that $x$ and $y$ are connected in $S$. This relation – "connected in $S$" – is an equivalence relation that partitions $S$ into its *connected components*. We say $S \subset \mathcal{X}$ is *connected* if it has a single connected component.

The cluster tree is a hierarchy each of whose levels is a partition of a *subset* of $\mathcal{X}$, which we will occasionally call a *subpartition* of $\mathcal{X}$. Write $\Pi(\mathcal{X}) = \{\text{subpartitions of } \mathcal{X}\}$.

**Definition 1** *For any $f : \mathcal{X} \to \mathbb{R}$, the* cluster tree *of $f$ is a function $\mathbb{C}_f : \mathbb{R} \to \Pi(\mathcal{X})$ given by*

$$\mathbb{C}_f(\lambda) = \text{connected components of } \{x \in \mathcal{X} : f(x) \geq \lambda\}.$$

*Any element of $\mathbb{C}_f(\lambda)$, for any $\lambda$, is called a* cluster *of $f$.*

For any $\lambda$, $\mathbb{C}_f(\lambda)$ is a set of disjoint clusters of $\mathcal{X}$. They form a hierarchy in the following sense.

**Lemma 2** *Pick any $\lambda' \leq \lambda$. Then:*

1. *For any $C \in \mathbb{C}_f(\lambda)$, there exists $C' \in \mathbb{C}_f(\lambda')$ such that $C \subseteq C'$.*

2. *For any $C \in \mathbb{C}_f(\lambda)$ and $C' \in \mathbb{C}_f(\lambda')$, either $C \subseteq C'$ or $C \cap C' = \emptyset$.*

We will sometimes deal with the restriction of the cluster tree to a finite set of points $x_1, \ldots, x_n$. Formally, the restriction of a subpartition $\mathbb{C} \in \Pi(\mathcal{X})$ to these points is defined to be $\mathbb{C}[x_1, \ldots, x_n] = \{C \cap \{x_1, \ldots, x_n\} : C \in \mathbb{C}\}$. Likewise, the restriction of the cluster tree is $\mathbb{C}_f[x_1, \ldots, x_n] : \mathbb{R} \to \Pi(\{x_1, \ldots, x_n\})$, where $\mathbb{C}_f[x_1, \ldots, x_n](\lambda) = \mathbb{C}_f(\lambda)[x_1, \ldots, x_n]$. See Figure 2 for an example.

## 2.2 Notion of convergence and previous work

Suppose a sample $X_n \subset \mathcal{X}$ of size $n$ is used to construct a tree $\mathbb{C}_n$ that is an estimate of $\mathbb{C}_f$. Hartigan [5] provided a very natural notion of consistency for this setting.

**Definition 3** *For any sets $A, A' \subset \mathcal{X}$, let $A_n$ (resp, $A'_n$) denote the smallest cluster of $\mathbb{C}_n$ containing $A \cap X_n$ (resp, $A' \cap X_n$). We say $\mathbb{C}_n$ is* consistent *if, whenever $A$ and $A'$ are different connected components of $\{x : f(x) \geq \lambda\}$ (for some $\lambda > 0$), $\mathbb{P}(A_n$ is disjoint from $A'_n) \to 1$ as $n \to \infty$.*

It is well known that if $X_n$ is used to build a uniformly consistent density estimate $f_n$ (that is, $\sup_x |f_n(x) - f(x)| \to 0$), then the cluster tree $\mathbb{C}_{f_n}$ is consistent; see the appendix for details. The big problem is that $\mathbb{C}_{f_n}$ is not easy to compute for typical density estimates $f_n$: imagine, for instance, how one might go about trying to find level sets of a mixture of Gaussians! Wong and

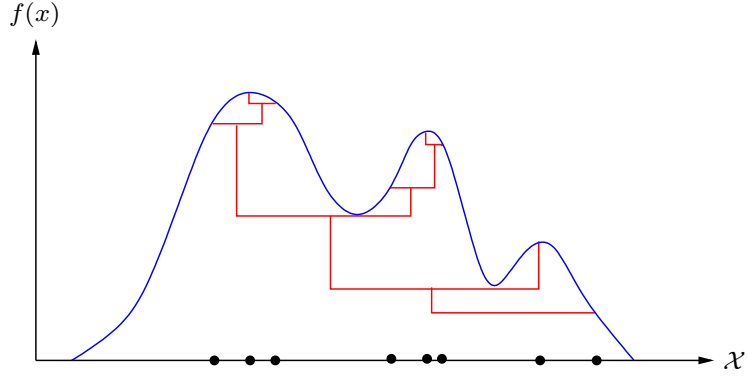

Figure 2: A probability density $f$, and the restriction of $\mathbb{C}_f$ to a finite set of eight points.

Lane [14] have an efficient procedure that tries to approximate $\mathbb{C}_{f_n}$ when $f_n$ is a $k$-nearest neighbor density estimate, but they have not shown that it preserves the consistency property of $\mathbb{C}_{f_n}$.

There is a simple and elegant algorithm that is a plausible estimator of the cluster tree: *single linkage* (or *Kruskal's algorithm*); see the appendix for pseudocode. Hartigan [5] has shown that it is consistent in one dimension ($d = 1$). But he also demonstrates, by a lovely reduction to continuum percolation, that this consistency fails in higher dimension $d \geq 2$. The problem is the requirement that $A \cap X_n \subset A_n$: by the time the clusters are large enough that one of them contains all of $A$, there is a reasonable chance that this cluster will be so big as to also contain part of $A'$.

With this insight, Hartigan defines a weaker notion of *fractional consistency*, under which $A_n$ (resp, $A'_n$) need not contain *all* of $A \cap X_n$ (resp, $A' \cap X_n$), but merely a sizeable chunk of it – and ought to be very close (at distance $\to 0$ as $n \to \infty$) to the remainder. He then shows that single linkage has this weaker consistency property for any pair $A, A'$ for which the ratio of $\inf\{f(x) : x \in A \cup A'\}$ to $\sup\{\inf\{f(x) : x \in P\} :$ paths $P$ from $A$ to $A'\}$ is sufficiently large. More recent work by Penrose [7] closes the gap and shows fractional consistency whenever this ratio is $> 1$.

A more robust version of single linkage has been proposed by Wishart [13]: when connecting points at distance $r$ from each other, only consider points that have at least $k$ neighbors within distance $r$ (for some $k > 2$). Thus initially, when $r$ is small, only the regions of highest density are available for linkage, while the rest of the data set is ignored. As $r$ gets larger, more and more of the data points become candidates for linkage. This scheme is intuitively sensible, but Wishart does not provide a proof of convergence. Thus it is unclear how to set $k$, for instance.

Stuetzle and Nugent [12] have an appealing top-down scheme for estimating the cluster tree, along with a post-processing step (called *runt pruning*) that helps identify modes of the distribution. The consistency of this method has not yet been established.

Several recent papers [6, 10, 9, 11] have considered the problem of recovering the connected components of $\{x : f(x) \geq \lambda\}$ for a user-specified $\lambda$: the *flat* version of our problem. In particular, the algorithm of [6] is intuitively similar to ours, though they use a single graph in which each point is connected to its $k$ nearest neighbors, whereas we have a hierarchy of graphs in which each point is connected to other points at distance $\leq r$ (for various $r$). Interestingly, $k$-nn graphs are valuable for flat clustering because they can adapt to clusters of different scales (different average interpoint distances). But they are challenging to analyze and seem to require various regularity assumptions on the data. A pleasant feature of the hierarchical setting is that different scales appear at different levels of the tree, rather than being collapsed together. This allows the use of $r$-neighbor graphs, and makes possible an analysis that has minimal assumptions on the data.

## 3  Algorithm and results

In this paper, we consider a generalization of Wishart's scheme and of single linkage, shown in Figure 3. It has two free parameters: $k$ and $\alpha$. For practical reasons, it is of interest to keep these as

---

1. For each $x_i$ set $r_k(x_i) = \inf\{r : B(x_i, r) \text{ contains } k \text{ data points}\}$.
2. As $r$ grows from $0$ to $\infty$:
    (a) Construct a graph $G_r$ with nodes $\{x_i : r_k(x_i) \le r\}$.
        Include edge $(x_i, x_j)$ if $\|x_i - x_j\| \le \alpha r$.
    (b) Let $\widehat{\mathbb{C}}(r)$ be the connected components of $G_r$.

---

Figure 3: Algorithm for hierarchical clustering. The input is a sample $X_n = \{x_1, \ldots, x_n\}$ from density $f$ on $\mathcal{X}$. Parameters $k$ and $\alpha$ need to be set. Single linkage is $(\alpha = 1, k = 2)$. Wishart suggested $\alpha = 1$ and larger $k$.

small as possible. We provide finite-sample convergence rates for all $1 \le \alpha \le 2$ and we can achieve $k \sim d \log n$, which we conjecture to be the best possible, if $\alpha > \sqrt{2}$. Our rates for $\alpha = 1$ force $k$ to be much larger, exponential in $d$. It is a fascinating open problem to determine whether the setting $(\alpha = 1, k \sim d \log n)$ yields consistency.

## 3.1 A notion of cluster salience

Suppose density $f$ is supported on some subset $\mathcal{X}$ of $\mathbb{R}^d$. We will show that the hierarchical clustering procedure is consistent in the sense of Definition 3. But the more interesting question is, what clusters will be identified from a *finite* sample? To answer this, we introduce a notion of salience.

The first consideration is that a cluster is hard to identify if it contains a thin "bridge" that would make it look disconnected in a small sample. To control this, we consider a "buffer zone" of width $\sigma$ around the clusters.

**Definition 4** *For $Z \subset \mathbb{R}^d$ and $\sigma > 0$, write $Z_\sigma = Z + B(0, \sigma) = \{y \in \mathbb{R}^d : \inf_{z \in Z} \|y - z\| \le \sigma\}$.*

An important technical point is that $Z_\sigma$ is a full-dimensional set, even if $Z$ itself is not.

Second, the ease of distinguishing two clusters $A$ and $A'$ depends inevitably upon the separation between them. To keep things simple, we'll use the same $\sigma$ as a separation parameter.

**Definition 5** *Let $f$ be a density on $\mathcal{X} \subset \mathbb{R}^d$. We say that $A, A' \subset \mathcal{X}$ are $(\sigma, \epsilon)$-separated if there exists $S \subset \mathcal{X}$ (separator set) such that:*

- *Any path in $\mathcal{X}$ from $A$ to $A'$ intersects $S$.*

- $\sup_{x \in S_\sigma} f(x) < (1 - \epsilon) \inf_{x \in A_\sigma \cup A'_\sigma} f(x)$.

Under this definition, $A_\sigma$ and $A'_\sigma$ must lie within $\mathcal{X}$, otherwise the right-hand side of the inequality is zero. However, $S_\sigma$ need not be contained in $\mathcal{X}$.

## 3.2 Consistency and finite-sample rate of convergence

Here we state the result for $\alpha > \sqrt{2}$ and $k \sim d \log n$. The analysis section also has results for $1 \le \alpha \le 2$ and $k \sim (2/\alpha)^d d \log n$.

**Theorem 6** *There is an absolute constant $C$ such that the following holds. Pick any $\delta, \epsilon > 0$, and run the algorithm on a sample $X_n$ of size $n$ drawn from $f$, with settings*

$$\sqrt{2}\left(1 + \frac{\epsilon^2}{\sqrt{d}}\right) \le \alpha \le 2 \quad and \quad k = C \cdot \frac{d \log n}{\epsilon^2} \cdot \log^2 \frac{1}{\delta}.$$

*Then there is a mapping $r : [0, \infty) \to [0, \infty)$ such that with probability at least $1 - \delta$, the following holds uniformly for all pairs of connected subsets $A, A' \subset \mathcal{X}$: If $A, A'$ are $(\sigma, \epsilon)$-separated (for $\epsilon$ and some $\sigma > 0$), and if*

$$\lambda := \inf_{x \in A_\sigma \cup A'_\sigma} f(x) \ge \frac{1}{v_d(\sigma/2)^d} \cdot \frac{k}{n} \cdot \left(1 + \frac{\epsilon}{2}\right) \tag{*}$$

*where $v_d$ is the volume of the unit ball in $\mathbb{R}^d$, then:*

1. *Separation. $A \cap X_n$ is disconnected from $A' \cap X_n$ in $G_{r(\lambda)}$.*

2. *Connectedness. $A \cap X_n$ and $A' \cap X_n$ are each individually connected in $G_{r(\lambda)}$.*

The two parts of this theorem – separation and connectedness – are proved in Sections 3.3 and 3.4.

We mention in passing that this finite-sample result implies consistency (Definition 3): as $n \to \infty$, take $k_n = (d \log n)/\epsilon_n^2$ with any schedule of $(\epsilon_n : n = 1, 2, \ldots)$ such that $\epsilon_n \to 0$ and $k_n/n \to 0$. Under mild conditions, any two connected components $A, A'$ of $\{f \geq \lambda\}$ are $(\sigma, \epsilon)$-separated for some $\sigma, \epsilon > 0$ (see appendix); thus they will get distinguished for sufficiently large $n$.

### 3.3 Analysis: separation

The cluster tree algorithm depends heavily on the radii $r_k(x)$: the distance within which $x$'s nearest $k$ neighbors lie (including $x$ itself). Thus the empirical probability mass of $B(x, r_k(x))$ is $k/n$. To show that $r_k(x)$ is meaningful, we need to establish that the mass of this ball under density $f$ is also, very approximately, $k/n$. The uniform convergence of these empirical counts follows from the fact that balls in $\mathbb{R}^d$ have finite VC dimension, $d + 1$. Using uniform Bernstein-type bounds, we get a set of basic inequalities that we use repeatedly.

**Lemma 7** *Assume $k \geq d \log n$, and fix some $\delta > 0$. Then there exists a constant $C_\delta$ such that with probability $> 1 - \delta$, every ball $B \subset \mathbb{R}^d$ satisfies the following conditions:*

$$f(B) \geq \frac{C_\delta d \log n}{n} \quad \implies \quad f_n(B) > 0$$

$$f(B) \geq \frac{k}{n} + \frac{C_\delta}{n}\sqrt{kd \log n} \quad \implies \quad f_n(B) \geq \frac{k}{n}$$

$$f(B) \leq \frac{k}{n} - \frac{C_\delta}{n}\sqrt{kd \log n} \quad \implies \quad f_n(B) < \frac{k}{n}$$

*Here $f_n(B) = |X_n \cap B|/n$ is the empirical mass of $B$, while $f(B) = \int_B f(x)dx$ is its true mass.*

PROOF: See appendix. $C_\delta = 2C_o \log(2/\delta)$, where $C_o$ is the absolute constant from Lemma 16. $\square$

We will henceforth think of $\delta$ as fixed, so that we do not have to repeatedly quantify over it.

**Lemma 8** *Pick $0 < r < 2\sigma/(\alpha + 2)$ such that*

$$\begin{aligned} v_d r^d \lambda &\geq \frac{k}{n} + \frac{C_\delta}{n}\sqrt{kd \log n} \\ v_d r^d \lambda(1 - \epsilon) &< \frac{k}{n} - \frac{C_\delta}{n}\sqrt{kd \log n} \end{aligned}$$

*(recall that $v_d$ is the volume of the unit ball in $\mathbb{R}^d$). Then with probability $> 1 - \delta$:*

1. *$G_r$ contains all points in $(A_{\sigma-r} \cup A'_{\sigma-r}) \cap X_n$ and no points in $S_{\sigma-r} \cap X_n$.*

2. *$A \cap X_n$ is disconnected from $A' \cap X_n$ in $G_r$.*

PROOF: For (1), any point $x \in (A_{\sigma-r} \cup A'_{\sigma-r})$ has $f(B(x, r)) \geq v_d r^d \lambda$; and thus, by Lemma 7, has at least $k$ neighbors within radius $r$. Likewise, any point $x \in S_{\sigma-r}$ has $f(B(x, r)) < v_d r^d \lambda(1 - \epsilon)$; and thus, by Lemma 7, has strictly fewer than $k$ neighbors within distance $r$.

For (2), since points in $S_{\sigma-r}$ are absent from $G_r$, any path from $A$ to $A'$ in that graph must have an edge across $S_{\sigma-r}$. But any such edge has length at least $2(\sigma - r) > \alpha r$ and is thus not in $G_r$. $\square$

**Definition 9** *Define $r(\lambda)$ to be the value of $r$ for which $v_d r^d \lambda = \frac{k}{n} + \frac{C_\delta}{n}\sqrt{kd \log n}$.*

To satisfy the conditions of Lemma 8, it suffices to take $k \geq 4C_\delta^2(d/\epsilon^2) \log n$; this is what we use.

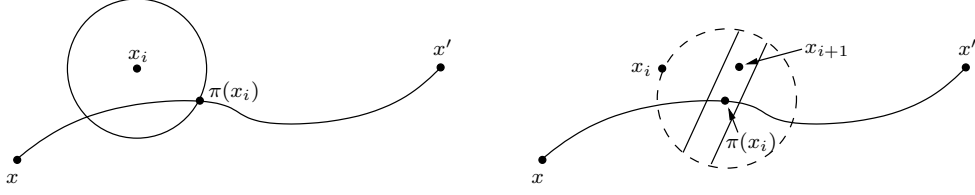

Figure 4: *Left:* $P$ is a path from $x$ to $x'$, and $\pi(x_i)$ is the point furthest along the path that is within distance $r$ of $x_i$. *Right:* The next point, $x_{i+1} \in X_n$, is chosen from a slab of $B(\pi(x_i), r)$ that is perpendicular to $x_i - \pi(x_i)$ and has width $2\zeta/\sqrt{d}$.

### 3.4 Analysis: connectedness

We need to show that points in $A$ (and similarly $A'$) are connected in $G_{r(\lambda)}$. First we state a simple bound (proved in the appendix) that works if $\alpha = 2$ and $k \sim d \log n$; later we consider smaller $\alpha$.

**Lemma 10** *Suppose* $1 \leq \alpha \leq 2$. *Then with probability* $\geq 1 - \delta$, $A \cap X_n$ *is connected in* $G_r$ *whenever* $r \leq 2\sigma/(2 + \alpha)$ *and the conditions of Lemma 8 hold, and*

$$v_d r^d \lambda \geq \left( \frac{2}{\alpha} \right)^d \frac{C_\delta d \log n}{n}.$$

Comparing this to the definition of $r(\lambda)$, we see that choosing $\alpha = 1$ would entail $k \geq 2^d$, which is undesirable. We can get a more reasonable setting of $k \sim d \log n$ by choosing $\alpha = 2$, but we'd like $\alpha$ to be as small as possible. A more refined argument shows that $\alpha \approx \sqrt{2}$ is enough.

**Theorem 11** *Suppose* $\alpha^2 \geq 2(1 + \zeta/\sqrt{d})$, *for some* $0 < \zeta \leq 1$. *Then, with probability* $> 1 - \delta$, $A \cap X_n$ *is connected in* $G_r$ *whenever* $r \leq \sigma/2$ *and the conditions of Lemma 8 hold, and*

$$v_d r^d \lambda \geq \frac{8}{\zeta} \cdot \frac{C_\delta d \log n}{n}.$$

PROOF: We have already made heavy use of uniform convergence over balls. We now also require a more complicated class $\mathcal{G}$, each element of which is the *intersection* of an open ball and a slab defined by two parallel hyperplanes. Formally, each of these functions is defined by a center $\mu$ and a unit direction $u$, and is the indicator function of the set

$$\{z \in \mathbb{R}^d : \|z - \mu\| < r, |(z - \mu) \cdot u| \leq \zeta r/\sqrt{d}\}.$$

We will describe any such set as "the slab of $B(\mu, r)$ in direction $u$". A simple calculation (see Lemma 4 of [4]) shows that the volume of this slab is at least $\zeta/4$ that of $B(x, r)$. Thus, if the slab lies entirely in $A_\sigma$, its probability mass is at least $(\zeta/4)v_d r^d \lambda$. By uniform convergence over $\mathcal{G}$ (which has VC dimension $2d$), we can then conclude (as in Lemma 7) that if $(\zeta/4)v_d r^d \lambda \geq (2C_\delta d \log n)/n$, then with probability at least $1 - \delta$, every such slab within $A$ contains at least one data point.

Pick any $x, x' \in A \cap X_n$; there is a path $P$ in $A$ with $x \overset{P}{\rightsquigarrow} x'$. We'll identify a sequence of data points $x_0 = x, x_1, x_2, \ldots$, ending in $x'$, such that for every $i$, point $x_i$ is active in $G_r$ and $\|x_i - x_{i+1}\| \leq \alpha r$. This will confirm that $x$ is connected to $x'$ in $G_r$.

To begin with, recall that $P$ is a continuous $1 - 1$ function from $[0, 1]$ into $A$. We are also interested in the inverse $P^{-1}$, which sends a point on the path back to its parametrization in $[0, 1]$. For any point $y \in \mathcal{X}$, define $N(y)$ to be the portion of $[0, 1]$ whose image under $P$ lies in $B(y, r)$: that is, $N(y) = \{0 \leq z \leq 1 : P(z) \in B(y, r)\}$. If $y$ is within distance $r$ of $P$, then $N(y)$ is nonempty. Define $\pi(y) = P(\sup N(y))$, the furthest point along the path within distance $r$ of $y$ (Figure 4, left).

The sequence $x_0, x_1, x_2, \ldots$ is defined iteratively; $x_0 = x$, and for $i = 0, 1, 2, \ldots$:

- If $\|x_i - x'\| \leq \alpha r$, set $x_{i+1} = x'$ and stop.

- By construction, $x_i$ is within distance $r$ of path $P$ and hence $N(x_i)$ is nonempty.
- Let $B$ be the open ball of radius $r$ around $\pi(x_i)$. The slab of $B$ in direction $x_i - \pi(x_i)$ must contain a data point; this is $x_{i+1}$ (Figure 4, right).

The process eventually stops because each $\pi(x_{i+1})$ is strictly further along path $P$ than $\pi(x_i)$; formally, $P^{-1}(\pi(x_{i+1})) > P^{-1}(\pi(x_i))$. This is because $\|x_{i+1} - \pi(x_i)\| < r$, so by continuity of the function $P$, there are points further along the path (beyond $\pi(x_i)$) whose distance to $x_{i+1}$ is still $< r$. Thus $x_{i+1}$ is distinct from $x_0, x_1, \ldots, x_i$. Since there are finitely many data points, the process must terminate, so the sequence $\{x_i\}$ does constitute a path from $x$ to $x'$.

Each $x_i$ lies in $A_r \subseteq A_{\sigma-r}$ and is thus active in $G_r$ (Lemma 8). Finally, the distance between successive points is:

$$
\begin{aligned}
\|x_i - x_{i+1}\|^2 &= \|x_i - \pi(x_i) + \pi(x_i) - x_{i+1}\|^2 \\
&= \|x_i - \pi(x_i)\|^2 + \|\pi(x_i) - x_{i+1}\|^2 + 2(x_i - \pi(x_i)) \cdot (\pi(x_i) - x_{i+1}) \\
&\leq 2r^2 + \frac{2\zeta r^2}{\sqrt{d}} \leq \alpha^2 r^2,
\end{aligned}
$$

where the second-last inequality comes from the definition of slab. $\square$

To complete the proof of Theorem 6, take $k = 4C_\delta^2(d/\epsilon^2)\log n$, which satisfies the requirements of Lemma 8 as well as those of Theorem 11, using $\zeta = 2\epsilon^2$. The relationship that defines $r(\lambda)$ (Definition 9) then translates into

$$
v_d r^d \lambda = \frac{k}{n}\left(1 + \frac{\epsilon}{2}\right).
$$

This shows that clusters at density level $\lambda$ emerge when the growing radius $r$ of the cluster tree algorithm reaches roughly $(k/(\lambda v_d n))^{1/d}$. In order for $(\sigma, \epsilon)$-separated clusters to be distinguished, we need this radius to be at most $\sigma/2$; this is what yields the final lower bound on $\lambda$.

# 4  Lower bound

We have shown that the algorithm of Figure 3 distinguishes pairs of clusters that are $(\sigma, \epsilon)$-separated. The number of samples it requires to capture clusters at density $\geq \lambda$ is, by Theorem 6,

$$
O\left(\frac{d}{v_d(\sigma/2)^d \lambda \epsilon^2}\log\frac{d}{v_d(\sigma/2)^d \lambda \epsilon^2}\right),
$$

We'll now show that this dependence on $\sigma$, $\lambda$, and $\epsilon$ is optimal. The only room for improvement, therefore, is in constants involving $d$.

**Theorem 12** *Pick any $\epsilon$ in $(0, 1/2)$, any $d > 1$, and any $\sigma, \lambda > 0$ such that $\lambda v_{d-1}\sigma^d < 1/50$. Then there exist: an input space $\mathcal{X} \subset \mathbb{R}^d$; a finite family of densities $\Theta = \{\theta_i\}$ on $\mathcal{X}$; subsets $A_i, A_i', S_i \subset \mathcal{X}$ such that $A_i$ and $A_i'$ are $(\sigma, \epsilon)$-separated by $S_i$ for density $\theta_i$, and $\inf_{x \in A_{i,\sigma} \cup A_{i,\sigma}'} \theta_i(x) \geq \lambda$, with the following additional property.*

*Consider any algorithm that is given $n \geq 100$ i.i.d. samples $X_n$ from some $\theta_i \in \Theta$ and, with probability at least $1/2$, outputs a tree in which the smallest cluster containing $A_i \cap X_n$ is disjoint from the smallest cluster containing $A_i' \cap X_n$. Then*

$$
n = \Omega\left(\frac{1}{v_d \sigma^d \lambda \epsilon^2 d^{1/2}}\log\frac{1}{v_d \sigma^d \lambda}\right).
$$

PROOF: We start by constructing the various spaces and densities. $\mathcal{X}$ is made up of two disjoint regions: a cylinder $\mathcal{X}_0$, and an additional region $\mathcal{X}_1$ whose sole purpose is as a repository for excess probability mass. Let $B_{d-1}$ be the unit ball in $\mathbb{R}^{d-1}$, and let $\sigma B_{d-1}$ be this same ball scaled to have radius $\sigma$. The cylinder $\mathcal{X}_0$ stretches along the $x_1$-axis; its cross-section is $\sigma B_{d-1}$ and its length is $4(c+1)\sigma$ for some $c > 1$ to be specified: $\mathcal{X}_0 = [0, 4(c+1)\sigma] \times \sigma B_{d-1}$. Here is a picture of it:

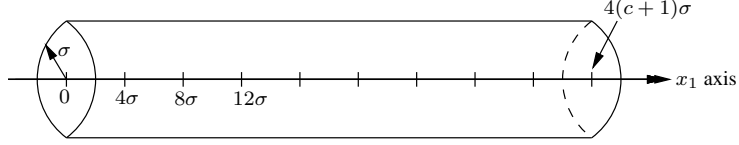

We will construct a family of densities $\Theta = \{\theta_i\}$ on $\mathcal{X}$, and then argue that any cluster tree algorithm that is able to distinguish $(\sigma, \epsilon)$-separated clusters must be able, when given samples from some $\theta_I$, to determine the identity of $I$. The sample complexity of this latter task can be lower-bounded using Fano's inequality (typically stated as in [2], but easily rewritten in the convenient form of [15], see appendix): it is $\Omega((\log |\Theta|)/\beta)$, for $\beta = \max_{i \neq j} K(\theta_i, \theta_j)$, where $K(\cdot, \cdot)$ is KL divergence.

The family $\Theta$ contains $c - 1$ densities $\theta_1, \ldots, \theta_{c-1}$, where $\theta_i$ is defined as follows:

- Density $\lambda$ on $[0, 4\sigma i + \sigma] \times \sigma B_{d-1}$ and on $[4\sigma i + 3\sigma, 4(c+1)\sigma] \times \sigma B_{d-1}$. Since the cross-sectional area of the cylinder is $v_{d-1}\sigma^{d-1}$, the total mass here is $\lambda v_{d-1}\sigma^d(4(c+1) - 2)$.
- Density $\lambda(1 - \epsilon)$ on $(4\sigma i + \sigma, 4\sigma i + 3\sigma) \times \sigma B_{d-1}$.
- Point masses $1/(2c)$ at locations $4\sigma, 8\sigma, \ldots, 4c\sigma$ along the $x_1$-axis (use arbitrarily narrow spikes to avoid discontinuities).
- The remaining mass, $1/2 - \lambda v_{d-1}\sigma^d(4(c+1) - 2\epsilon)$, is placed on $\mathcal{X}_1$ in some fixed manner (that does not vary between different densities in $\Theta$).

Here is a sketch of $\theta_i$. The low-density region of width $2\sigma$ is centered at $4\sigma i + 2\sigma$ on the $x_1$-axis.

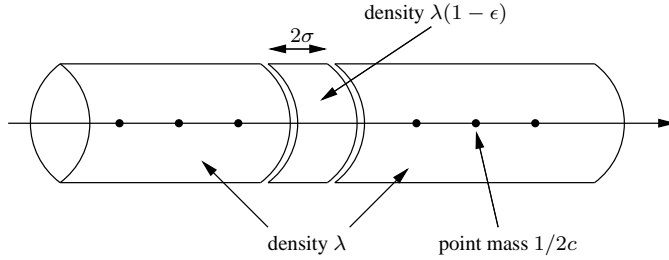

For any $i \neq j$, the densities $\theta_i$ and $\theta_j$ differ only on the cylindrical sections $(4\sigma i + \sigma, 4\sigma i + 3\sigma) \times \sigma B_{d-1}$ and $(4\sigma j + \sigma, 4\sigma j + 3\sigma) \times \sigma B_{d-1}$, which are disjoint and each have volume $2v_{d-1}\sigma^d$. Thus

$$
\begin{aligned}
K(\theta_i, \theta_j) &= 2v_{d-1}\sigma^d \left( \lambda \log \frac{\lambda}{\lambda(1-\epsilon)} + \lambda(1-\epsilon) \log \frac{\lambda(1-\epsilon)}{\lambda} \right) \\
&= 2v_{d-1}\sigma^d \lambda(-\epsilon \log(1-\epsilon)) \leq \frac{4}{\ln 2} v_{d-1}\sigma^d \lambda \epsilon^2
\end{aligned}
$$

(using $\ln(1-x) \geq -2x$ for $0 < x \leq 1/2$). This is an upper bound on the $\beta$ in the Fano bound.

Now define the clusters and separators as follows: for each $1 \leq i \leq c - 1$,

- $A_i$ is the line segment $[\sigma, 4\sigma i]$ along the $x_1$-axis,
- $A_i'$ is the line segment $[4\sigma(i+1), 4(c+1)\sigma - \sigma]$ along the $x_1$-axis, and
- $S_i = \{4\sigma i + 2\sigma\} \times \sigma B_{d-1}$ is the cross-section of the cylinder at location $4\sigma i + 2\sigma$.

Thus $A_i$ and $A_i'$ are one-dimensional sets while $S_i$ is a $(d-1)$-dimensional set. It can be checked that $A_i$ and $A_i'$ are $(\sigma, \epsilon)$-separated by $S_i$ in density $\theta_i$.

With the various structures defined, what remains is to argue that if an algorithm is given a sample $X_n$ from some $\theta_I$ (where $I$ is unknown), and is able to separate $A_I \cap X_n$ from $A_I' \cap X_n$, then it can effectively infer $I$. This has sample complexity $\Omega((\log c)/\beta)$. Details are in the appendix. $\square$

There remains a discrepancy of $2^d$ between the upper and lower bounds; it is an interesting open problem to close this gap. Does the $(\alpha = 1, k \sim d \log n)$ setting (yet to be analyzed) do the job?

**Acknowledgments.** We thank the anonymous reviewers for their detailed and insightful comments, and the National Science Foundation for support under grant IIS-0347646.

# References

[1] O. Bousquet, S. Boucheron, and G. Lugosi. Introduction to statistical learning theory. *Lecture Notes in Artificial Intelligence*, 3176:169–207, 2004.

[2] T. Cover and J. Thomas. *Elements of Information Theory*. Wiley, 2005.

[3] S. Dasgupta and Y. Freund. Random projection trees for vector quantization. *IEEE Transactions on Information Theory*, 55(7):3229–3242, 2009.

[4] S. Dasgupta, A. Kalai, and C. Monteleoni. Analysis of perceptron-based active learning. *Journal of Machine Learning Research*, 10:281–299, 2009.

[5] J.A. Hartigan. Consistency of single linkage for high-density clusters. *Journal of the American Statistical Association*, 76(374):388–394, 1981.

[6] M. Maier, M. Hein, and U. von Luxburg. Optimal construction of k-nearest neighbor graphs for identifying noisy clusters. *Theoretical Computer Science*, 410:1749–1764, 2009.

[7] M. Penrose. Single linkage clustering and continuum percolation. *Journal of Multivariate Analysis*, 53:94–109, 1995.

[8] D. Pollard. Strong consistency of k-means clustering. *Annals of Statistics*, 9(1):135–140, 1981.

[9] P. Rigollet and R. Vert. Fast rates for plug-in estimators of density level sets. *Bernoulli*, 15(4):1154–1178, 2009.

[10] A. Rinaldo and L. Wasserman. Generalized density clustering. *Annals of Statistics*, 38(5):2678–2722, 2010.

[11] A. Singh, C. Scott, and R. Nowak. Adaptive hausdorff estimation of density level sets. *Annals of Statistics*, 37(5B):2760–2782, 2009.

[12] W. Stuetzle and R. Nugent. A generalized single linkage method for estimating the cluster tree of a density. *Journal of Computational and Graphical Statistics*, 19(2):397–418, 2010.

[13] D. Wishart. Mode analysis: a generalization of nearest neighbor which reduces chaining effects. In *Proceedings of the Colloquium on Numerical Taxonomy held in the University of St. Andrews*, pages 282–308, 1969.

[14] M.A. Wong and T. Lane. A kth nearest neighbour clustering procedure. *Journal of the Royal Statistical Society Series B*, 45(3):362–368, 1983.

[15] B. Yu. Assouad, Fano and Le Cam. *Festschrift for Lucien Le Cam*, pages 423–435, 1997.

